# Semi-supervised Learning by Entropy Minimization

**Yves Grandvalet** *
Heudiasyc, CNRS/UTC
60205 Compiègne cedex, France
grandval@utc.fr

**Yoshua Bengio**
Dept. IRO, Université de Montréal
Montreal, Qc, H3C 3J7, Canada
bengioy@iro.umontreal.ca

## Abstract

We consider the semi-supervised learning problem, where a decision rule is to be learned from labeled and unlabeled data. In this framework, we motivate minimum entropy regularization, which enables to incorporate unlabeled data in the standard supervised learning. Our approach includes other approaches to the semi-supervised problem as particular or limiting cases. A series of experiments illustrates that the proposed solution benefits from unlabeled data. The method challenges mixture models when the data are sampled from the distribution class spanned by the generative model. The performances are definitely in favor of minimum entropy regularization when generative models are misspecified, and the weighting of unlabeled data provides robustness to the violation of the "cluster assumption". Finally, we also illustrate that the method can also be far superior to manifold learning in high dimension spaces.

## 1  Introduction

In the classical supervised learning classification framework, a decision rule is to be learned from a learning set $\mathcal{L}_n = \{\mathbf{x}_i, y_i\}_{i=1}^n$, where each example is described by a pattern $\mathbf{x}_i \in \mathcal{X}$ and by the supervisor's response $y_i \in \Omega = \{\omega_1, \ldots, \omega_K\}$. We consider semi-supervised learning, where the supervisor's responses are limited to a subset of $\mathcal{L}_n$.

In the terminology used here, semi-supervised learning refers to learning a decision rule on $\mathcal{X}$ from labeled and unlabeled data. However, the related problem of transductive learning, i.e. of predicting labels on a set of predefined patterns, is addressed as a side issue. Semi-supervised problems occur in many applications where labeling is performed by human experts. They have been receiving much attention during the last few years, but some important issues are unresolved [10].

In the probabilistic framework, semi-supervised learning can be modeled as a missing data problem, which can be addressed by generative models such as mixture models thanks to the EM algorithm and extensions thereof [6].Generative models apply to the joint density of patterns and class $(X, Y)$. They have appealing features, but they also have major drawbacks. Their estimation is much more demanding than discriminative models, since the model of $P(X, Y)$ is exhaustive, hence necessarily more complex than the model of

$P(Y|X)$. More parameters are to be estimated, resulting in more uncertainty in the estimation process. The generative model being more precise, it is also more likely to be misspecified. Finally, the fitness measure is not discriminative, so that better models are not necessarily better predictors of class labels. These difficulties have lead to proposals aiming at processing unlabeled data in the framework of supervised classification [1, 5, 11]. Here, we propose an estimation principle applicable to any probabilistic classifier, aiming at making the most of unlabeled data when they are beneficial, while providing a control on their contribution to provide robustness to the learning scheme.

## 2 Derivation of the Criterion

### 2.1 Likelihood

We first recall how the semi-supervised learning problem fits into standard supervised learning by using the maximum (conditional) likelihood estimation principle. The learning set is denoted $\mathcal{L}_n = \{\mathbf{x}_i, \mathbf{z}_i\}_{i=1}^n$, where $\mathbf{z} \in \{0,1\}^K$ denotes the dummy variable representing the actually available labels (while $y$ represents the precise and complete class information): if $\mathbf{x}_i$ is labeled $\omega_k$, then $\mathbf{z}_{ik} = 1$ and $\mathbf{z}_{i\ell} = 0$ for $\ell \neq k$; if $\mathbf{x}_i$ is unlabeled, then $\mathbf{z}_{i\ell} = 1$ for $\ell = 1, \ldots, K$.

We assume that labeling is missing at random, that is, for all unlabeled examples, $P(\mathbf{z}|\mathbf{x}, \omega_k) = P(\mathbf{z}|\mathbf{x}, \omega_\ell)$, for any $(\omega_k, \omega_\ell)$ pair, which implies

$$P(\omega_k|\mathbf{x}, \mathbf{z}) = \frac{z_k P(\omega_k|\mathbf{x})}{\sum_{\ell=1}^K z_\ell P(\omega_\ell|\mathbf{x})} \quad . \tag{1}$$

Assuming independent examples, the conditional log-likelihood of $(Z|X)$ on the observed sample is then

$$L(\boldsymbol{\theta}; \mathcal{L}_n) = \sum_{i=1}^n \log \left( \sum_{k=1}^K z_{ik} f_k(\mathbf{x}_i; \boldsymbol{\theta}) \right) + h(\mathbf{z}_i) \quad , \tag{2}$$

where $h(\mathbf{z})$, which does not depend on $P(X, Y)$, is only affected by the missingness mechanism, and $f_k(\mathbf{x}; \boldsymbol{\theta})$ is the model of $P(\omega_k|\mathbf{x})$ parameterized by $\boldsymbol{\theta}$.

This criterion is a concave function of $f_k(\mathbf{x}_i; \boldsymbol{\theta})$, and for simple models such as the ones provided by logistic regression, it is also concave in $\boldsymbol{\theta}$, so that the global solution can be obtained by numerical optimization. Maximizing (2) corresponds to maximizing the complete likelihood if no assumption whatsoever is made on $P(X)$ [6].

Provided $f_k(\mathbf{x}_i; \boldsymbol{\theta})$ sum to one, the likelihood is not affected by unlabeled data: unlabeled data convey no information. In the maximum a posteriori (MAP) framework, Seeger remarks that unlabeled data are useless regarding discrimination when the priors on $P(X)$ and $P(Y|X)$ factorize [10]: observing $\mathbf{x}$ does not inform about $y$, unless the modeler assumes so. Benefitting from unlabeled data requires assumptions of some sort on the relationship between $X$ and $Y$. In the Bayesian framework, this will be encoded by a prior distribution. As there is no such thing like a universally relevant prior, we should look for an induction bias exploiting unlabeled data when the latter is known to convey information.

### 2.2 When Are Unlabeled Examples Informative?

Theory provides little support to the numerous experimental evidences [5, 7, 8] showing that unlabeled examples can help the learning process. Learning theory is mostly developed at the two extremes of the statistical paradigm: in parametric statistics where examples are known to be generated from a known class of distribution, and in the distribution-free Structural Risk Minimization (SRM) or Probably Approximately Correct (PAC) frameworks. Semi-supervised learning, in the terminology used here, does not fit the distribution-free frameworks: no positive statement can be made without distributional assumptions, as for

some distributions $P(X, Y)$ unlabeled data are non-informative while supervised learning is an easy task. In this regard, generalizing from labeled and unlabeled data may differ from transductive inference.

In parametric statistics, theory has shown the benefit of unlabeled examples, either for specific distributions [9], or for mixtures of the form $P(\mathbf{x}) = pP(\mathbf{x}|\omega_1) + (1-p)P(\mathbf{x}|\omega_2)$ where the estimation problem is essentially reduced to the one of estimating the mixture parameter $p$ [4]. These studies conclude that the (asymptotic) information content of unlabeled examples decreases as classes overlap.[1] Thus, the assumption that classes are well separated is sensible if we expect to take advantage of unlabeled examples.

The conditional entropy $H(Y|X)$ is a measure of class overlap, which is invariant to the parameterization of the model. This measure is related to the usefulness of unlabeled data where labeling is indeed ambiguous. Hence, we will measure the conditional entropy of class labels conditioned on the observed variables

$$H(Y|X, Z) = -E_{XYZ}[\log P(Y|X, Z)] \ , \tag{3}$$

where $E_X$ denotes the expectation with respect to $X$.

In the Bayesian framework, assumptions are encoded by means of a prior on the model parameters. Stating that we expect a high conditional entropy does not uniquely define the form of the prior distribution, but the latter can be derived by resorting to the maximum entropy principle.[2] Let $(\boldsymbol{\theta}, \boldsymbol{\psi})$ denote the model parameters of $P(X, Y, Z)$; the maximum entropy prior verifying $E_{\Theta\Psi}[H(Y|X, Z)] = c$, where the constant $c$ quantifies how small the entropy should be on average, takes the form

$$P(\boldsymbol{\theta}, \boldsymbol{\psi}) \propto \exp\left(-\lambda H(Y|X, Z))\right) \ , \tag{4}$$

where $\lambda$ is the positive Lagrange multiplier corresponding to the constant $c$.

Computing $H(Y|X, Z)$ requires a model of $P(X, Y, Z)$ whereas the choice of the diagnosis paradigm is motivated by the possibility to limit modeling to conditional probabilities. We circumvent the need of additional modeling by applying the plug-in principle, which consists in replacing the expectation with respect to $(X, Z)$ by the sample average. This substitution, which can be interpreted as "modeling" $P(X, Z)$ by its empirical distribution, yields

$$H_{\text{emp}}(Y|X, Z; \mathcal{L}_n) = -\frac{1}{n}\sum_{i=1}^{n}\sum_{k=1}^{K} P(\omega_k|\mathbf{x}_i, \mathbf{z}_i) \log P(\omega_k|\mathbf{x}_i, \mathbf{z}_i) \ . \tag{5}$$

This empirical functional is plugged in (4) to define an empirical prior on parameters $\boldsymbol{\theta}$, that is, a prior whose form is partly defined from data [2].

## 2.3 Entropy Regularization

Recalling that $f_k(\mathbf{x}; \boldsymbol{\theta})$ denotes the model of $P(\omega_k|\mathbf{x})$, the model of $P(\omega_k|\mathbf{x}, \mathbf{z})$ (1) is defined as follows:

$$g_k(\mathbf{x}, \mathbf{z}; \boldsymbol{\theta}) = \frac{z_k f_k(\mathbf{x}; \boldsymbol{\theta})}{\sum_{\ell=1}^{K} z_\ell f_\ell(\mathbf{x}; \boldsymbol{\theta})} \ .$$

For labeled data, $g_k(\mathbf{x}, \mathbf{z}; \boldsymbol{\theta}) = z_k$, and for unlabeled data, $g_k(\mathbf{x}, \mathbf{z}; \boldsymbol{\theta}) = f_k(\mathbf{x}; \boldsymbol{\theta})$.
From now on, we drop the reference to parameter $\boldsymbol{\theta}$ in $f_k$ and $g_k$ to lighten notation. The

MAP estimate is the maximizer of the posterior distribution, that is, the maximizer of

$$
\begin{aligned}
C(\boldsymbol{\theta}, \lambda; \mathcal{L}_n) &= L(\boldsymbol{\theta}; \mathcal{L}_n) - \lambda H_{\mathrm{emp}}(Y|X, Z; \mathcal{L}_n) \\
&= \sum_{i=1}^{n} \log \left( \sum_{k=1}^{K} z_{ik} f_k(\mathbf{x}_i) \right) + \lambda \sum_{i=1}^{n} \sum_{k=1}^{K} g_k(\mathbf{x}_i, \mathbf{z}_i) \log g_k(\mathbf{x}_i, \mathbf{z}_i) \quad , (6)
\end{aligned}
$$

where the constant terms in the log-likelihood (2) and log-prior (4) have been dropped. While $L(\boldsymbol{\theta}; \mathcal{L}_n)$ is only sensitive to labeled data, $H_{\mathrm{emp}}(Y|X, Z; \mathcal{L}_n)$ is only affected by the value of $f_k(\mathbf{x})$ on unlabeled data.

Note that the approximation $H_{\mathrm{emp}}$ (5) of $H$ (3) breaks down for wiggly functions $f_k(\cdot)$ with abrupt changes between data points (where $P(X)$ is bounded from below). As a result, it is important to constrain $f_k(\cdot)$ in order to enforce the closeness of the two functionals. In the following experimental section, we imposed a smoothness constraint on $f_k(\cdot)$ by adding to the criterion $C$ (6) a penalizer with its corresponding Lagrange multiplier $\nu$.

## 3 Related Work

**Self-Training**   Self-training [7] is an iterative process, where a learner imputes the labels of examples which have been classified with confidence in the previous step. Amini *et al.* [1] analyzed this technique and shown that it is equivalent to a version of the classification EM algorithm, which minimizes the likelihood deprived of the entropy of the partition. In the context of conditional likelihood with labeled and unlabeled examples, the criterion is

$$
\sum_{i=1}^{n} \log \left( \sum_{k=1}^{K} z_{ik} f_k(\mathbf{x}_i) \right) + \sum_{k=1}^{K} g_k(\mathbf{x}_i) \log g_k(\mathbf{x}_i) \quad ,
$$

which is recognized as an instance of the criterion (6) with $\lambda = 1$.

Self-confident logistic regression [5] is another algorithm optimizing the criterion for $\lambda = 1$. Using smaller $\lambda$ values is expected to have two benefits: first, the influence of unlabeled examples can be controlled, in the spirit of the EM-$\lambda$ [8], and second, slowly increasing $\lambda$ defines a scheme similar to deterministic annealing, which should help the optimization process to avoid poor local minima of the criterion.

**Minimum entropy methods**   Minimum entropy regularizers have been used in other contexts to encode learnability priors (e.g. [3]). In a sense, $H_{\mathrm{emp}}$ can be seen as a poor's man way to generalize this approach to continuous input spaces. This empirical functional was also used by Zhu *et al.* [13, Section 6] as a criterion to learn weight function parameters in the context of transduction on manifolds for learning.

**Input-Dependent Regularization**   Our criterion differs from input-dependent regularization [10, 11] in that it is expressed only in terms of $P(Y|X, Z)$ and does not involve $P(X)$. However, we stress that for unlabeled data, the regularizer agrees with the complete likelihood provided $P(X)$ is small near the decision surface. Indeed, whereas a generative model would maximize $\log P(X)$ on the unlabeled data, our criterion minimizes the conditional entropy on the same points. In addition, when the model is regularized (e.g. with weight decay), the conditional entropy is prevented from being too small close to the decision surface. This will favor putting the decision surface in a low density area.

## 4 Experiments
### 4.1 Artificial Data

In this section, we chose a simple experimental setup in order to avoid artifacts stemming from optimization problems. Our goal is to check to what extent supervised learning can be improved by unlabeled examples, and if minimum entropy can compete with generative models which are usually advocated in this framework.

The minimum entropy regularizer is applied to the logistic regression model. It is compared to logistic regression fitted by maximum likelihood (ignoring unlabeled data) and logistic regression with all labels known. The former shows what has been gained by handling unlabeled data, and the latter provides the "crystal ball" performance obtained by guessing correctly all labels. All hyper-parameters (weight-decay for all logistic regression models plus the $\lambda$ parameter (6) for minimum entropy) are tuned by ten-fold cross-validation.

Minimum entropy logistic regression is also compared to the classic EM algorithm for Gaussian mixture models (two means and one common covariance matrix estimated by maximum likelihood on labeled and unlabeled examples, see e.g. [6]). Bad local maxima of the likelihood function are avoided by initializing EM with the parameters of the true distribution when the latter is a Gaussian mixture, or with maximum likelihood parameters on the (fully labeled) test sample when the distribution departs from the model. This initialization advantages EM, since it is guaranteed to pick, among all local maxima of the likelihood, the one which is in the basin of attraction of the optimal value. Furthermore, this initialization prevents interferences that may result from the "pseudo-labels" given to unlabeled examples at the first E-step. In particular, "label switching" (i.e. badly labeled clusters) is avoided at this stage.

**Correct joint density model**   In the first series of experiments, we consider two-class problems in an 50-dimensional input space. Each class is generated with equal probability from a normal distribution. Class $\omega_1$ is normal with mean $(aa \ldots a)$ and unit covariance matrix. Class $\omega_2$ is normal with mean $-(aa \ldots a)$ and unit covariance matrix. Parameter $a$ tunes the Bayes error which varies from 1 % to 20 % (1 %, 2.5 %, 5 %, 10 %, 20 %). The learning sets comprise $n_l$ labeled examples, $(n_l = 50, 100, 200)$ and $n_u$ unlabeled examples, $(n_u = n_l \times (1, 3, 10, 30, 100))$. Overall, 75 different setups are evaluated, and for each one, 10 different training samples are generated. Generalization performances are estimated on a test set of size 10 000.

This benchmark provides a comparison for the algorithms in a situation where unlabeled data are known to convey information. Besides the favorable initialization of the EM algorithm to the optimal parameters, EM benefits from the *correctness* of the model: data were generated according to the model, that is, two Gaussian subpopulations with identical covariances. The logistic regression model is only *compatible* with the joint distribution, which is a weaker fulfillment than correctness.

As there is no modeling bias, differences in error rates are only due to differences in estimation efficiency. The overall error rates (averaged over all settings) are in favor of minimum entropy logistic regression ($14.1 \pm 0.3$ %). EM ($15.6 \pm 0.3$ %) does worse on average than logistic regression ($14.9 \pm 0.3$ %). For reference, the average Bayes error rate is 7.7 % and logistic regression reaches $10.4 \pm 0.1$ % when all examples are labeled.

Figure 1 provides more informative summaries than these raw numbers. The plots represent the error rates (averaged over $n_l$) versus Bayes error rate and the $n_u/n_l$ ratio. The first plot shows that, as asymptotic theory suggests [4, 9], unlabeled examples are mostly informative when the Bayes error is low. This observation validates the relevance of the minimum entropy assumption. This graph also illustrates the consequence of the demanding parametrization of generative models. Mixture models are outperformed by the simple logistic regression model when the sample size is low, since their number of parameters grows quadratically (*vs.* linearly) with the number of input features.

The second plot shows that the minimum entropy model takes quickly advantage of unlabeled data when classes are well separated. With $n_u = 3n_l$, the model considerably improves upon the one discarding unlabeled data. At this stage, the generative models do not perform well, as the number of available examples is low compared to the number of parameters in the model. However, for very large sample sizes, with 100 times more unla-

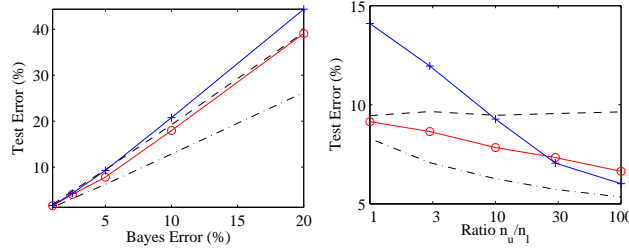

Figure 1: Left: test error *vs.* Bayes error rate for $n_u/n_l = 10$; right: test error *vs.* $n_u/n_l$ ratio for 5 % Bayes error ($a = 0.23$). Test errors of minimum entropy logistic regression ($\circ$) and mixture models ($+$). The errors of logistic regression (dashed), and logistic regression with all labels known (dash-dotted) are shown for reference.

beled examples than labeled examples, the generative approach eventually becomes more accurate than the diagnosis approach.

**Misspecified joint density model** In a second series of experiments, the setup is slightly modified by letting the class-conditional densities be corrupted by outliers. For each class, the examples are generated from a mixture of two Gaussians centered on the same mean: a unit variance component gathers 98 % of examples, while the remaining 2 % are generated from a large variance component, where each variable has a standard deviation of 10. The mixture model used by EM is slightly misspecified since it is a simple Gaussian mixture. The results, displayed in the left-hand-side of Figure 2, should be compared with the right-hand-side of Figure 1. The generative model dramatically suffers from the misspecification and behaves worse than logistic regression for all sample sizes. The unlabeled examples have first a beneficial effect on test error, then have a detrimental effect when they overwhelm the number of labeled examples. On the other hand, the diagnosis models behave smoothly as in the previous case, and the minimum entropy criterion performance improves.

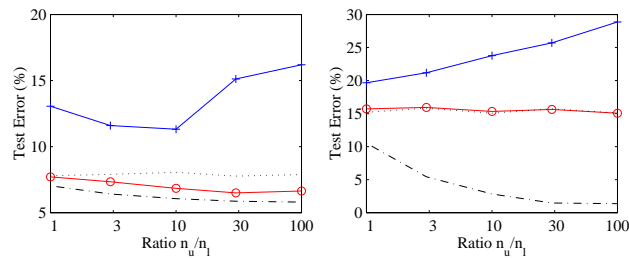

Figure 2: Test error *vs.* $n_u/n_l$ ratio for $a = 0.23$. Average test errors for minimum entropy logistic regression ($\circ$) and mixture models ($+$). The test error rates of logistic regression (dotted), and logistic regression with all labels known (dash-dotted) are shown for reference. Left: experiment with outliers; right: experiment with uninformative unlabeled data.

The last series of experiments illustrate the robustness with respect to the cluster assumption, by testing it on distributions where unlabeled examples are not informative, and where a low density $P(X)$ does not indicate a boundary region. The data is drawn from two Gaussian clusters like in the first series of experiment, but the label is now independent of the clustering: an example $\mathbf{x}$ belongs to class $\omega_1$ if $x_2 > x_1$ and belongs to class $\omega_2$ otherwise:

the Bayes decision boundary is now separates each cluster in its middle. The mixture model is unchanged. It is now far from the model used to generate data. The right-hand-side plot of Figure 1 shows that the favorable initialization of EM does not prevent the model to be fooled by unlabeled data: its test error steadily increases with the amount of unlabeled data. On the other hand, the diagnosis models behave well, and the minimum entropy algorithm is not distracted by the two clusters; its performance is nearly identical to the one of training with labeled data only (cross-validation provides $\lambda$ values close to zero), which can be regarded as the ultimate performance in this situation.

**Comparison with manifold transduction**  Although our primary goal is to infer a decision function, we also provide comparisons with a transduction algorithm of the "manifold family". We chose the consistency method of Zhou *et al.* [12] for its simplicity. As suggested by the authors, we set $\alpha = 0.99$ and the scale parameter $\sigma^2$ was optimized on test results [12]. The results are reported in Table 1. The experiments are limited due to the memory requirements of the consistency method in our naive MATLAB implementation.

Table 1: Error rates (%) of minimum entropy (ME) *vs.* consistency method (CM), for $a = 0.23$, $n_l = 50$, and a) pure Gaussian clusters b) Gaussian clusters corrupted by outliers c) class boundary separating one Gaussian cluster

| $n_u$ | 50 | 150 | 500 | 1500 |
|---|---|---|---|---|
| a) ME | $10.8 \pm 1.5$ | $9.8 \pm 1.9$ | $8.8 \pm 2.0$ | $8.3 \pm 2.6$ |
| a) CM | $21.4 \pm 7.2$ | $25.5 \pm 8.1$ | $29.6 \pm 9.0$ | $26.8 \pm 7.2$ |
| b) ME | $8.5 \pm 0.9$ | $8.3 \pm 1.5$ | $7.5 \pm 1.5$ | $6.6 \pm 1.5$ |
| b) CM | $22.0 \pm 6.7$ | $25.6 \pm 7.4$ | $29.8 \pm 9.7$ | $27.7 \pm 6.8$ |
| c) ME | $8.7 \pm 0.8$ | $8.3 \pm 1.1$ | $7.2 \pm 1.0$ | $7.2 \pm 1.7$ |
| c) CM | $51.6 \pm 7.9$ | $50.5 \pm 4.0$ | $49.3 \pm 2.6$ | $50.2 \pm 2.2$ |

The results are extremely poor for the consistency method, whose error is way above minimum entropy, and which does not show any sign of improvement as the sample of unlabeled data grows. Furthermore, when classes do not correspond to clusters, the consistency method performs random class assignments. In fact, our setup, which was designed for the comparison of global classifiers, is extremely defavorable to manifold methods, since the data is truly 50-dimensional. In this situation, local methods suffer from the "curse of dimensionality", and many more unlabeled examples would be required to get sensible results. Hence, these results mainly illustrate that manifold learning is not the best choice in semi-supervised learning for truly high dimensional data.

### 4.2   Facial Expression Recognition

We now consider an image recognition problem, consisting in recognizing seven (balanced) classes corresponding to the universal emotions (anger, fear, disgust, joy, sadness, surprise and neutral). The patterns are gray level images of frontal faces, with standardized positions. The data set comprises 375 such pictures made of $140 \times 100$ pixels.

We tested kernelized logistic regression (Gaussian kernel), its minimum entropy version, nearest neigbor and the consistency method. We repeatedly (10 times) sampled 1/10 of the dataset for providing the labeled part, and the remainder for testing. Although $(\alpha, \sigma^2)$ were chosen to minimize the test error, the consistency method performed poorly with $63.8 \pm 1.3$ % test error (compared to 86 % error for random assignments). Nearest-neighbor get similar results with $63.1 \pm 1.3$ % test error, and Kernelized logistic regression (ignoring unlabeled examples) improved to reach $53.6 \pm 1.3$ %. Minimum entropy kernelized logistic regression regression achieves $52.0 \pm 1.9$ % error (compared to about 20 % errors for human on this database). The scale parameter chosen for kernelized logistic regression (by ten-fold cross-validation) amount to use a global classifier. Again, the local methods

fail. This may be explained by the fact that the database contains several pictures of each person, with different facial expressions. Hence, local methods are likely to pick the same identity instead of the same expression, while global methods are able to learn the relevant directions.

## 5 Discussion

We propose to tackle the semi-supervised learning problem in the supervised learning framework by using the minimum entropy regularizer. This regularizer is motivated by theory, which shows that unlabeled examples are mostly beneficial when classes have small overlap. The MAP framework provides a means to control the weight of unlabeled examples, and thus to depart from optimism when unlabeled data tend to harm classification.

Our proposal encompasses self-learning as a particular case, as minimizing entropy increases the confidence of the classifier output. It also approaches the solution of transductive large margin classifiers in another limiting case, as minimizing entropy is a means to drive the decision boundary from learning examples.

The minimum entropy regularizer can be applied to both local and global classifiers. As a result, it can improve over manifold learning when the dimensionality of data is effectively high, that is, when data do not lie on a low-dimensional manifold. Also, our experiments suggest that the minimum entropy regularization may be a serious contender to generative models. It compares favorably to these mixture models in three situations: for small sample sizes, where the generative model cannot completely benefit from the knowledge of the correct joint model; when the joint distribution is (even slightly) misspecified; when the unlabeled examples turn out to be non-informative regarding class probabilities.

## Footnotes

*This work was supported in part by the IST Programme of the European Community, under the PASCAL Network of Excellence IST-2002-506778. This publication only reflects the authors' views.

[1]This statement, given explicitly by [9], is also formalized, though not stressed, by [4], where the Fisher information for unlabeled examples at the estimate $\hat{p}$ is clearly a measure of the overlap between class conditional densities: $I_u(\hat{p}) = \int \frac{(P(\mathbf{x}|\omega_1) - P(\mathbf{x}|\omega_2))^2}{\hat{p}P(\mathbf{x}|\omega_1) + (1-\hat{p})P(\mathbf{x}|\omega_2)} d\mathbf{x}$.

[2]Here, maximum entropy refers to the construction principle which enables to derive distributions from constraints, not to the content of priors regarding entropy.

## References

[1] M. R. Amini and P. Gallinari. Semi-supervised logistic regression. In *15th European Conference on Artificial Intelligence*, pages 390–394. IOS Press, 2002.

[2] J. O. Berger. *Statistical Decision Theory and Bayesian Analysis*. Springer, New York, 2 edition, 1985.

[3] M. Brand. Structure learning in conditional probability models via an entropic prior and parameter extinction. *Neural Computation*, 11(5):1155–1182, 1999.

[4] V. Castelli and T. M. Cover. The relative value of labeled and unlabeled samples in pattern recognition with an unknown mixing parameter. *IEEE Trans. on Information Theory*, 42(6):2102–2117, 1996.

[5] Y. Grandvalet. Logistic regression for partial labels. In *9th Information Processing and Management of Uncertainty in Knowledge-based Systems – IPMU'02*, pages 1935–1941, 2002.

[6] G. J. McLachlan. *Discriminant analysis and statistical pattern recognition*. Wiley, 1992.

[7] K. Nigam and R. Ghani. Analyzing the effectiveness and applicability of co-training. In *Ninth International Conference on Information and Knowledge Management*, pages 86–93, 2000.

[8] K. Nigam, A. K. McCallum, S. Thrun, and T. Mitchell. Text classification from labeled and unlabeled documents using EM. *Machine learning*, 39(2/3):135–167, 2000.

[9] T. J. O'Neill. Normal discrimination with unclassified observations. *Journal of the American Statistical Association*, 73(364):821–826, 1978.

[10] M. Seeger. Learning with labeled and unlabeled data. Technical report, Institute for Adaptive and Neural Computation, University of Edinburgh, 2002.

[11] M. Szummer and T. S. Jaakkola. Information regularization with partially labeled data. In *Advances in Neural Information Processing Systems 15*. MIT Press, 2003.

[12] D. Zhou, O. Bousquet, T. Navin Lal, J. Weston, and B. Schölkopf. Learning with local and global consistency. In *Advances in Neural Information Processing Systems 16*, 2004.

[13] X. Zhu, Z. Ghahramani, and J. Lafferty. Semi-supervised learning using Gaussian fields and harmonic functions. In *20th Int. Conf. on Machine Learning*, pages 912–919, 2003.
